# Automated State Abstraction for Options using the U-Tree Algorithm

**Anders Jonsson, Andrew G. Barto**
Department of Computer Science
University of Massachusetts
Amherst, MA 01003
{ajonsson,barto}@cs.umass.edu

## Abstract

Learning a complex task can be significantly facilitated by defining a hierarchy of subtasks. An agent can learn to choose between various temporally abstract actions, each solving an assigned subtask, to accomplish the overall task. In this paper, we study hierarchical learning using the framework of options. We argue that to take full advantage of hierarchical structure, one should perform option-specific state abstraction, and that if this is to scale to larger tasks, state abstraction should be automated. We adapt McCallum's U-Tree algorithm to automatically build option-specific representations of the state feature space, and we illustrate the resulting algorithm using a simple hierarchical task. Results suggest that automated option-specific state abstraction is an attractive approach to making hierarchical learning systems more effective.

## 1  Introduction

Researchers in the field of reinforcement learning have recently focused considerable attention on temporally abstract actions (e.g., [1, 3, 5, 6, 7, 9]). The term *temporally abstract* describes actions that can take variable amounts of time. One motivation for using temporally abstract actions is that they can be used to exploit the hierarchical structure of a problem. Among other things, a hierarchical structure is a natural way to incorporate prior knowledge into a learning system by allowing reuse of temporally abstract actions whose policies were learned in other tasks. Learning in a hierarchy can also significantly reduce the number of situations between which a learning agent needs to discriminate.

We use the framework of *options* [6, 9], which extends the theory of reinforcement learning to include temporally abstract actions. In many cases, accurately executing an option's policy does not depend on all state features available to the learning agent. Further, the features that are relevant often differ from option to option. Within a hierarchical learning system, it is possible to perform *option-specific state abstraction* by which irrelevant features specific to each option are ignored. Using option-specific state abstraction in a hierarchical learning system can save memory through the development of compact state representations, and it can accelerate learning because of the generalization induced by the abstraction.

Dietterich [2] introduced action-specific state abstraction in a hierarchy of temporally abstract actions. However, his approach requires the system developer to define a set of relevant state features for each action prior to learning. As the complexity of a problem grows, it becomes increasingly difficult to hand-code such state representations. One way to remedy this problem is to use an automated process for constructing state representations.

We apply McCallum's U-Tree algorithm [4] to individual options to achieve automated, option-specific state abstraction. The U-Tree algorithm automatically builds a state-feature representation starting from one that makes no distinctions between different observation vectors. Thus, no specification of state-feature dependencies is necessary prior to learning.

In Section 2, we give a brief description of the U-Tree algorithm. Section 3 introduces modifications necessary to make the U-Tree algorithm suitable for learning in a hierarchical system. We describe the setup of our experiments in Section 4 and present the results in Section 5. Section 6 concludes with a discussion of future work.

## 2 The U-Tree algorithm

The U-Tree algorithm [4] retains a history of transition instances $T_t = < T_{t-1}, a_{t-1}, r_t, s_t >$ composed of the observation vector, $s_t$, at time step $t$, the previous action, $a_{t-1}$, the reward, $r_t$, received during the transition into $s_t$, and the previous instance, $T_{t-1}$. A decision tree—the U-Tree—sorts a new instance $T_t$ based on its components and assigns it to a unique leaf of the tree. The distinctions associated with a leaf are determined by the root-to-leaf path.

For each leaf-action pair $(L_j, a)$, the algorithm keeps an action value $Q(L_j, a)$ estimating the future discounted reward associated with being in $L_j$ and executing $a$. The utility of a leaf is denoted $U(L_j) = \max_a Q(L_j, a)$. The algorithm also keeps a model consisting of estimated transition probabilities $\Pr(L_k|L_j, a)$ and expected immediate rewards $R(L_j, a)$ computed from the transition instances. The model is used in performing one sweep of value iteration after the execution of each action, modifying the values of all leaf-action pairs $(L_j, a)$:

$$Q(L_j, a) \leftarrow R(L_j, a) + \gamma \sum_{L_k} \Pr(L_k|L_j, a) U(L_k).$$

One can use other reinforcement learning algorithms to update the action values, such as Q-learning or prioritized sweeping.

The U-Tree algorithm periodically adds new distinctions to the tree in the form of temporary nodes, called fringe nodes, and performs statistical tests to see whether the added distinctions increase the predictive power of the U-Tree. Each distinction is based on (1) a perceptual dimension, which is either an observation or a previous action, and (2) a history index, indicating how far back in the current history the dimension will be examined. Each leaf of the tree is extended with a subtree of a fixed depth, $z$, constructed from permutations of all distinctions not already on the path to the leaf. The instances associated with the leaf are distributed to the leaves of the added subtree—the fringe nodes—according to the corresponding distinctions. A statistical test, the Kolmogorov-Smirnov (KS) test, compares the distributions of future discounted reward of the leaf node's policy action with that of a fringe node's policy action. The distribution of future discounted reward associated with a node $L_j$ and its policy action $a = \arg\max_a Q(L_j, a)$ is composed of the estimated future discounted reward of individual instances $T_t \in T(L_j, a)$ given by:

$$V(T_t) = r_{t+1} + \gamma \sum_{L_k} \Pr(L_k|L_j, a) U(L_k).$$

The KS test outputs a statistical difference $d_{L_j, L_k} \in [0, 1]$ between the distributions of two nodes $L_j$ and $L_k$. The U-Tree algorithm retains the subtree of distinctions $i$ at a leaf $L_j$

if the sum of the KS statistical differences over the fringe nodes $F(L_j,i)$ of the subtree is (1) larger than the sum of the KS differences of all other subtrees, and (2) exceeds some threshold $\theta$. That is, the tree is extended from leaf $L_j$ with a subtree $i$ of new distinctions if for all subtrees $m \neq i$:

$$\sum_{F(L_j,i)} d_{L_j,F(L_j,i)} > \sum_{F(L_j,m)} d_{L_j,F(L_j,m)}$$

$$\text{and} \quad \sum_{F(L_j,i)} d_{L_j,F(L_j,i)} > \theta.$$

Whenever the tree is extended, the action values of the previous leaf node are passed on to the new leaf nodes.

One can restrict the number of distinctions an agent can make at any one time by imposing a limit on the depth of the U-Tree. The length of the history the algorithm needs to retain depends only on the tree size and not on the size of the overall state set. Consequently, the algorithm has the potential to scale well to large tasks.

In previous experiments, the U-Tree algorithm was able to learn a compact state representation together with a satisfactory policy in a complex driving task [4]. A version of the U-Tree algorithm suitable for continuous state spaces has also been developed and successfully used in robot soccer [10].

## 3 Adapting the U-Tree algorithm for options

We now turn to the issue of adapting the U-Tree algorithm for use with options and hierarchical learning architectures. Given a finite Markov decision process with state set $S$, an option $o = <I,\pi,\beta>$ consists of a set $I \subseteq S$ of states from which the option can be initiated, a closed-loop policy $\pi$ for the choice of actions, and a termination condition $\beta$ which, for each state, gives the probability that the option will terminate when that state is reached. Primitive actions generalize to options that always terminate after one time step. It is easy to define hierarchies of options in which the policy of an option can select other options. A local reward function can be associated with an option to facilitate learning the option's policy.

What makes the U-Tree algorithm so suitable for performing option-specific state abstraction is that a U-Tree simultaneously defines a state representation and a policy over this representation. With a separate U-Tree assigned to each option, the algorithm is able to perform state abstraction separately for each option while modifying its policy.

Because options at different levels of a hierarchy operate on different time scales, their transition instances must take different forms. To make our scheme work, we need to add a notion of temporal abstraction to the definition of a transition instance:

**Definition:** *A transition instance of an option $o$ has the form $T_t^o = <T_{t-k}^o, o_{t-k}, R_t, s_t>$, where $s_t$ is the observation vector at time step $t$, $o_{t-k}$ is the option previously executed by option $o$, terminating at time $t$ and with a duration $k$, $R_t = \sum_{i=1}^{k} \gamma^{i-1} r_{t-k+i}$ is the discounted sum of rewards received during the execution of $o_{t-k}$, and $T_{t-k}^o$ is the previous instance.*

Since options at one level in a hierarchy are executed one at a time, they will each experience a different sequence of transition instances. For the U-Tree algorithm to work under these conditions, *the U-Tree of each option has to keep its own history of instances* and base distinctions on these instances alone.

The U-Tree algorithm was developed for infinite-horizon tasks. Because an option terminates and may not be executed again for some time, its associated history will be made up of finite segments corresponding to separate executions of the option. The first transition

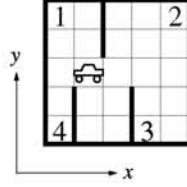

Figure 1: The Taxi task

instance recorded during an execution is independent of the last instance recorded during a previous execution. Consequently, we do not allow updates across segments. With these modifications, the U-Tree algorithm can be applied to hierarchical learning with options.

### 3.1 Intra-option learning

When several options operate in the same parts of the state space and choose from among the same actions, it is possible to learn something about one option from the behavior generated by the execution of other options. In a process called intra-option learning [8], the action values of one option are updated based on actions executed in another, associated option. The update only occurs if the action executed in the latter has a non-zero probability of being executed in the former.

Similarly, we can base distinctions in the U-Tree associated with one option on transition instances recorded during the execution of another option. We do this by adding instances recorded during the execution of one option to the history of each associated option. By associating each instance with a vector of leaves, one for the U-Tree of each option, this approach does not require additional memory for keeping multiple copies of an instance. For the scheme to work, we introduce a vector of rewards $\vec{R}_t = \{R_t^{o'}\}$ in an instance $T_t^o$, where $R_t^{o'}$ is the discounted sum of local rewards for each option $o'$ associated with $o_{t-k}$.

## 4  Experiments

We tested our version of the U-Tree algorithm on the Taxi task [1], in which an agent—the taxi—moves around on a grid (Figure 1). The taxi is assigned the task of delivering passengers from their locations to their destination, both chosen at random from the set of pick-up/drop-off sites $P = \{1, 2, 3, 4\}$. The taxi agent's observation vector $s = (x, y, l, d)$ is composed of the $(x, y)$-position of the taxi, the location $l \in P \cup \{\texttt{taxi}\}$ of the current passenger, and this passenger's destination $d \in P$. The actions available to the taxi are Pick-up, Drop-off, and Move($m$), $m \in \{\texttt{N}, \texttt{E}, \texttt{S}, \texttt{W}\}$, the four cardinal directions. When a passenger is delivered, a new passenger appears at a random pickup site. The rewards provided to the taxi are:

|  |  |
|---|---|
| 19 | for delivering the passenger |
| −11 | for illegal Pick-up or Drop-off |
| −1 | for any other action (including moving into walls) |

To aid the taxi agent we introduced four options: Navigate($p$) $= < I^p, \pi^p, \beta^p >$, $p \in P$, where, letting $S$ denote the set of all observation vectors and $G^p = \{(x, y, l, d) \in S \mid (x, y)$ is the location of $p\}$:

| | |
|---|---|
| $I^p$ : | $S - G^p$ |
| $\pi^p$ : | the policy for getting to $G^p$ that the agent is trying to learn |
| $\beta^p$ : | 1 if $s \in G^p$; 0 otherwise. |

We further introduced a local reward $R_t^p$ for Navigate($p$), identical to the global reward provided to the agent with the exception that $R_t^p = 9$ for reaching $G^p$.

In our application of the U-Tree algorithm to the taxi problem, the history of each option had a maximum length of 6,000 instances. If this length was exceeded, the oldest instance in the history was discarded. Expanding the tree was only considered if there were more than 3,000 instances in the history. We set the expansion depth $z$ to 1 and the expansion threshold $\theta$ to 1.0, except when no distinctions were present in the tree, in which case $\theta = 0.3$. The algorithm used this lower threshold when the agent was not able to make any distinctions because it is difficult in this case to accumulate enough evidence of statistical difference to accept a distinction. Since the U-Tree algorithm does not go back and reconsider distinctions in the tree, it is important to reduce the number of incorrect distinctions due to sparse statistical evidence. Therefore, our implementation only compared two distributions of future discounted reward between leaves if each contained more than 15 instances.

Because the taxi task is fully observable, we set the history index of the U-tree algorithm to zero. For exploration, the system used an $\varepsilon$-softmax strategy, which picks a random action with probability $\varepsilon$ and performs softmax otherwise. Normally, tuning the softmax temperature $\tau$ provides a good balance between exploration and exploitation, but as the U-Tree evolves, a new value of $\tau$ may improve performance. To avoid re-tuning $\tau$, the $\varepsilon$-random part ensured that all actions were executed regularly.

We designed one set of experiments to examine the efficiency of intra-option learning. We randomly selected one of the options Navigate($p$) to execute, and randomly selected a new position for the taxi whenever it reached $p$, ignoring the issue of delivering a passenger. At the beginning of each learning run, we assigned a U-Tree containing a single node to each option. In one set of runs, the algorithm used intra-option learning, and in another set, it used regular learning in which the U-Trees of different options did not share any instances.

In a second set of experiments, the policies of the options and the overall Taxi task were learned in parallel. We allowed the policy of the overall task to choose between the options Navigate($p$), and the actions Pick-up and Drop-off. The reward provided for the overall task was the sum of global reward and local reward of the option currently being executed (cf. Digney [3]). When a passenger was delivered, a new taxi position was selected randomly and a new passenger appeared at a randomly selected pickup site.

## 5   Results

The results from the intra-option learning experiments are shown in Figure 2. The graphs for intra-option learning (solid) and regular learning (broken) are averaged over 5 independent runs. We tuned $\tau$ and $\varepsilon$ for each set of learning runs to give maximum performance. At intervals of 500 time steps, the U-Trees of the options were saved and evaluated separately. The evaluation consisted of fixing a target, repeatedly navigating to that target for 25,000 time steps, randomly repositioning the taxi every time the target was reached, repeating for all targets, and adding the rewards. From these results, We conclude that (1) intra-option learning converges faster than regular learning, and (2) intra-option learning achieves a higher level of performance. Faster convergence is due to the fact that the histories associated with the options fill up more quickly during intra-option learning. Higher performance is achieved because the amount of evidence is larger. The target of an option is only reached once during each execution of the option, whereas it might be reached several times during the execution of another option.

In the second set of experiments, we performed 10 learning runs, each with a duration of

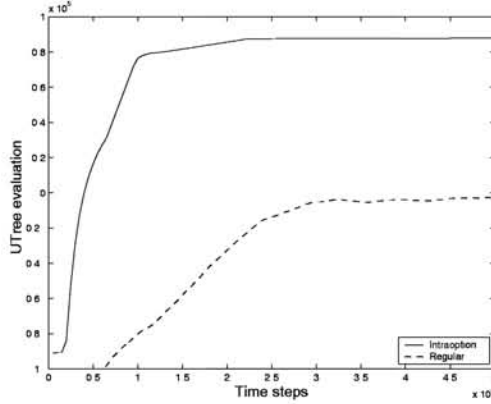

Figure 2: Comparison between intra-option and regular learning

200,000 time steps. Figure 3 shows an example of the resulting U-Trees. Nodes that represent distinctions are drawn as circles, and leaf nodes are shown as squares or, in most cases, omitted. In the figure, $a$ denotes a distinction over the previously executed option (in the order Navigate(p), Pick-up and Drop-off), and other letters denote a distinction over the corresponding observation. Note that the U-Tree of Navigate(1) did not make a distinction between $x$-positions in the lower part of the grid. In some places, for example in Navigate(4), the right branch of $x$, the algorithm made a suboptimal distinction. A distinction over $y$ would have given a smaller number of leaves and would have been sufficient to represent an optimal policy. The U-Trees in the figure contain a total of 188 leaf nodes. Across 10 runs, the number of leaf nodes varied from 154 to 259, with an average of 189. Some leaf nodes were never visited, making the actual number of states even smaller. This is comparable to the results of Dietterich [2] who hand-coded a representation containing 106 states. Compared to the 500 distinct states in a flat representation of the task, or the 2,500 distinct states that the five policies would require without abstraction, our result is a significant improvement. Certainly, the memory required to store histories should also be taken into account. However, we believe that the memory savings due to option-specific state abstraction in larger tasks will significantly outweigh the memory requirement for U-Trees.

## 6 Conclusion

We have shown that the U-Tree algorithm can be used with options in a hierarchical learning system. Our results suggest that automated option-specific state abstraction performed by the algorithm is an attractive approach to making hierarchical learning systems more effective. Although our testbed was small, we believe this is an important first step toward automated state abstraction in hierarchies. We also incorporated intra-option learning into the U-Tree algorithm, a method that allows a learning agent to extract more information from the training data. Results show that intra-option learning can significantly improve the performance of a learning agent performing option-specific state abstraction.

Although our main motivation for developing a hierarchical version of the U-Tree algorithm was automating state abstraction, the new definition of a transition instance enables history to be structured hierarchically, something that is useful when learning to solve problems in partially observable domains.

Future work will examine the performance of option-specific state abstraction using the U-Tree algorithm in larger, more realistic tasks. We also plan to develop a version of

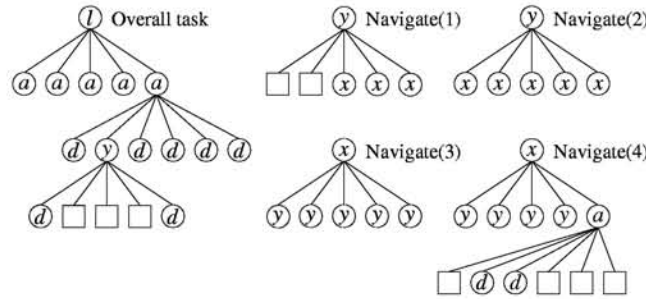

Figure 3: U-Trees for different policies

the U-Tree algorithm that goes back in the tree and reconsiders distinctions. This has the potential to improve the performance of the algorithm by correcting nodes for which incorrect distinctions were made.

## Acknowledgments

The authors would like to thank Tom Dietterich for providing code for the Taxi task, Andrew McCallum for valuable correspondence regarding the U-Tree algorithm, and Ted Perkins for reading and providing helpful comments on the paper. This work was funded by the National Science Foundation under Grant No. ECS-9980062. Any opinions, findings, and conclusions or recommendations expressed in this material are those of the authors and do not necessarily reflect the views of the National Science Foundation.

## References

[1] Dietterich, T. (2000). Hierarchical reinforcement learning with the MAXQ value function decomposition. *Artificial Intelligence Research* **13**:227–303.

[2] Dietterich, T. (2000) State Abstraction in MAXQ Hierarchical Reinforcement Learning. In S. A. Solla, T. K. Leen, and K.-R. Muller (eds.), *Advances in Neural Information Processing Systems 12*, pp. 994–1000. Cambridge MA: MIT Press.

[3] Digney, B. (1996) Emergent hierarchical control structures: Learning reactive/hierarchical relationships in reinforcement environments. In P. Meas and M. Mataric (eds.), *From animals to animats 4*. Cambridge MA: MIT Press.

[4] McCallum, A. (1995) *Reinforcement Learning with Selective Perception and Hidden State*. PhD thesis, Computer Science Department, University of Rochester.

[5] Parr, R., and Russell, S. (1998) Reinforcement learning with hierarchies of machines. In M. I. Jordan, M. J. Kearns, and S. A. Solla (eds.), *Advances in Neural Information Processing Systems 10*, pp. 1043–1049. Cambridge MA: MIT Press.

[6] Precup, D., and Sutton, R. (1998) Multi-time models for temporally abstract planning. In M. I. Jordan, M. J. Kearns, and S. A. Solla (eds.), *Advances in Neural Information Processing Systems 10*, pp. 1050–1056. Cambridge MA: MIT Press.

[7] Singh, S. (1992) Reinforcement learning with a hierarchy of abstract models. In *Proc. of the 10th National Conf. on Artificial Intelligence*, pp. 202–207. Menlo Park, CA: AAAI Press/MIT Press.

[8] Sutton, R., Precup, D., and Singh, S. (1998) Intra-Option Learning about Temporally Abstract Actions. In *Proc. of the 15th Intl. Conf. on Machine Learning, ICML'98*, pp. 556–564. Morgan Kaufman.

[9] Sutton, R., Precup, D., and Singh, S. (1999) Between MDPs and semi-MDPs: A framework for temporal abstraction in reinforcement learning. *Artificial Intelligence* **112**:181–211.

[10] Uther, W., and Veloso, M. (1997) Generalizing Adversarial Reinforcement Learning. *AAAI Fall Symposium on Model Directed Autonomous Systems*.
